# MCBoost: Multiple Classifier Boosting for Perceptual Co-clustering of Images and Visual Features

**Tae-Kyun Kim**[*]
Sidney Sussex College
University of Cambridge
Cambridge CB2 3HU, UK
tkk22@cam.ac.uk

**Roberto Cipolla**
Department of Engineering
University of Cambridge
Cambridge CB2 1PZ, UK
cipolla@cam.ac.uk

## Abstract

We present a new co-clustering problem of images and visual features. The problem involves a set of non-object images in addition to a set of object images and features to be co-clustered. Co-clustering is performed in a way that maximises discrimination of object images from non-object images, thus emphasizing discriminative features. This provides a way of obtaining *perceptual* joint-clusters of object images and features. We tackle the problem by simultaneously boosting multiple strong classifiers which compete for images by their expertise. Each boosting classifier is an aggregation of weak-learners, i.e. simple visual features. The obtained classifiers are useful for object detection tasks which exhibit multi-modalities, e.g. multi-category and multi-view object detection tasks. Experiments on a set of pedestrian images and a face data set demonstrate that the method yields intuitive image clusters with associated features and is much superior to conventional boosting classifiers in object detection tasks.

## 1 Introduction

It is known that visual cells (*visual features*) selectively respond to *imagery patterns* in perception. Learning process may be associated with co-clusters of visual features and imagery data in a way of facilitating image data perception. We formulate this in the context of boosting classifiers with simple visual features for object detection task [3]. There are two sets of images: a set of object images and a set of non-object images, labelled as positive and negative class members respectively. There are also a huge number of simple image features, only a small fraction of which are selected to discriminate the positive class from the negative class by $H(\mathbf{x}) = \sum_t \alpha_t h_t(\mathbf{x})$ where $\mathbf{x}$ is an input vector, $\alpha_t, h_t$ are the weight and the score of $t$-th weak-learner using a single feature. As object images typically exhibit multi-modalities, a single aggregation of simple features often does not dichotomise all object images from non-object images. Our problem is to find out subsets of object images, each of which is associated with a set of features for maximising classification. Note that image clusters to be obtained are coupled with selected features and likewise features to be selected are dependent on image clusters, requiring a concurrent clustering of images and features.

See Figure 1 for an example where subsets of face images are pose-wise obtained with associated features by the proposed method (Section 3). Features are placed around eyes, nose, mouth and etc. as the cues for discriminating faces from background. As such facial features are distributed differently mainly according to face pose, the obtained pose-wise face clusters are, therefore, intuitive and desirable in perception. Note the challenges in achieving this: The input set of face images are mixed up by different faces, lighting conditions as well as pose. Some are photographs of real-faces and the others are drawings. Desired image clusters are *not observable* in input space. See Figure 2

---

[*]Webpage: http://mi.eng.cam.ac.uk/~tkk22

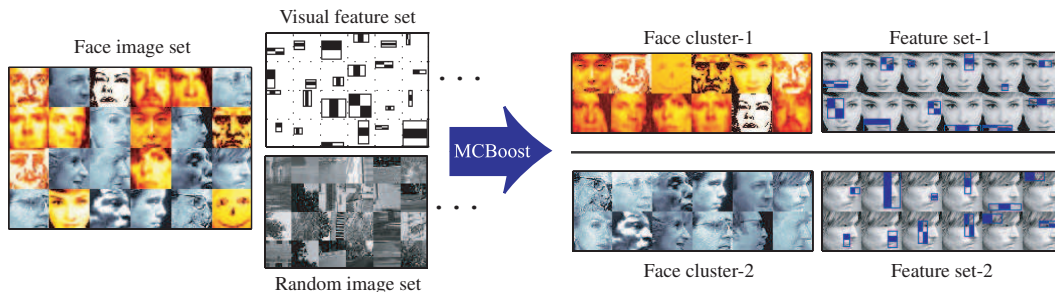

Figure 1: **Perceptual co-clusters of images and visual features.** *For given a set of face and random images and simple visual features, the proposed method finds perceptual joint-clusters of face images and features, which facilitates classification of face images from random images. Face clusters are pose-wise obtained.*

for the result of the traditional unsupervised method (k-means clustering) applied to the face images. Images of the obtained clusters are almost random with respect to pose. To obtain perceptual face clusters, a method requires a discriminative process and part-based representations (like the simple features used). Technically, we must be able to cope with an arbitrary initialisation of image clusters (as target clusters are hidden) and feature selection among a huge number of simple visual features.

The proposed method (Section 3) has potential for wide-applications in perceptual data exploration. It generally solves a new co-clustering problem of a data set (e.g. a set of face images) and a feature set (e.g. simple visual features) in a way to maximise discrimination of the data set from another data set (e.g. a set of random images). The method is also useful for object detection tasks. Boosting a classifier with simple features [3] is a state-of-the-art in object detection tasks. It delivers high accuracy and is very time-efficient. Conventionally, multiple boosting classifiers are separately learnt for multiple categories and/or multiple views of object images [6]. It is, however, tedious to manually label category/pose for a large data set and, importantly, it is not clear to define object categories and scopes of each pose. Would there be a better partitioning for learning multiple boost-

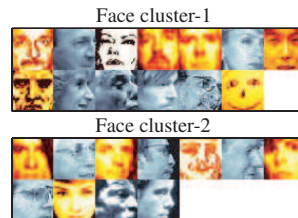

Figure 2: **Image sets obtained by the k-means clustering method.**

ing classifiers? We let this be a part of automatic learning in the proposed method. It simultaneously boosts multiple strong classifiers, each of which has expertise on a particular set of object images by a set of weak-learners.

The remainder of this paper is arranged as follows: we briefly review the previous work in Section 2 and present our solution in Section 3. Experiments and conclusions are drawn in Section 4 and Section 5 respectively.

## 2  Related work

Existing co-clustering work (e.g. [1]) is formulated as an unsupervised learning task. It simultaneously clusters rows and columns of a co-occurrence table by e.g. maximising mutual information between the cluster variables. Conversely, we make use of class labels for discriminative learning. Using a co-occurrence table in prior work is also prohibitive due to a huge number of visual features that we consider.

Mixture of Experts [2] (MoE) jointly learns multiple classifiers and data partitions. It much emphasises local experts and is suitable when input data can be naturally divided into homogeneous subsets, which is, however, often not possible as observed in Figure 2. In practice, it is difficult to establish a good initial data partition and to perform expert selection based on localities. Note that EM in MoE resorts to a local optimum. Furthermore, the data partitions of MoE could be undesirably affected by a large background class in our problem and the linear transformations used in MoE are limited for delivering a meaningful part-based representation of images.

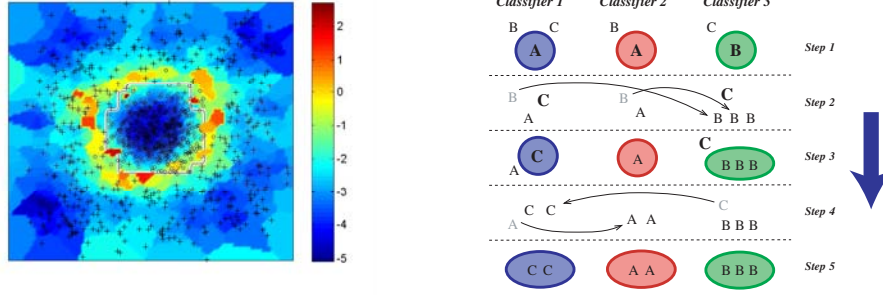

Figure 3: *(left) Risk map for given two class data (circle and cross). The weak-learners (either a vertical or horizontal line) found by Adaboost method [7] are placed on high risk regions. (right) State diagram for the concept of MCBoost.*

Boosting [7] is a sequential method of aggregating multiple (weak) classifiers. It finds weak-learners to correctly classify erroneous samples in previous weak-learners. While MoE makes a decision by dynamically selected local experts, all weak-learners contribute to a decision with learnt weights in boosting classifier. As afore-mentioned, expert selection is a difficult problem when an input space is not naturally divided into sub-regions (clusters). Boosting classifier solves various non-linear classification problems but cannot solve XOR problems where only half the data can be correctly classified by a set of weak-learners. Two disjointed sets of weak-learners, i.e. two boosting classifiers, are required to conquer each half of data by a set of weak-learners.

Torralba et al. have addressed joint-learning of multiple boosting classifiers for multiple category and multiple view object detection [4]. The complexity of resulting classifiers is reduced by sharing visual features among classifiers. Each classifier in their method is based on each of category-wise or pose-wise clusters of object images, which requires manual labels for cateogry/pose, whereas we optimise image clusters and boosting classifiers simultaneously.

## 3   MCBoost: multiple strong classifier boosting

Our formulation considers $K$ strong classifiers, each of which is represented by a linear combination of weak-learners as

$$H_k(\mathbf{x}) = \sum_t \alpha_{kt} h_{kt}(\mathbf{x}), \qquad k = 1, ... K, \tag{1}$$

where $\alpha_{kt}$ and $h_{kt}$ are the weight and the score of $t$-th weak-learner of $k$-th strong classifier. Each strong classifier is devoted to a subset of input patterns allowing repetition and each weak-learner in a classifier comprises of a single visual feature and a threshold. For aggregating multiple strong classifiers, we formulate Noisy-OR as

$$P(\mathbf{x}) = 1 - \prod_k (1 - P_k(\mathbf{x})), \tag{2}$$

where $P_k(\mathbf{x}) = \frac{1}{1+\exp(-H_k(\mathbf{x}))}$. It assigns samples to a positive class if any of classifiers does and assigns samples to a negative class if every classifier does. Conventional design in object detection study [6] also favours OR decision as it does not require classifier selection. An individual classifier is learnt from a subset of positive samples and all negative samples, enforcing a positive sample to be accepted by one of the classifiers and a negative sample to be rejected by all. Our derivation builds on the previous Noisy-OR Boost algorithm [5], which has been proposed for multiple instance learning.

The sample weights are initialised by random partitioning of positive samples, i.e. $w_{ki} = 1$ if $\mathbf{x}_i \in k$ and $w_{ki} = 0$ otherwise, where $i$ and $k$ denote $i$-th sample and $k$-th classifier respectively. We set $w_{ki} = 1/K$ for all $k$'s for negative samples. For given weights, the method finds $K$ weak-learners

---
**Algorithm 1.** MCBoost

    **Input:** A data set $(\mathbf{x}_i, y_i)$ and a set of pre-defined weak-learners
    **Output:** Multiple boosting classifiers $\mathrm{H}_k(\mathbf{x}) = \sum_{t=1}^T \alpha_{kt}\mathrm{h}_{kt}(\mathbf{x}), k = 1..., K$
---
1. Compute a reduced set of weak-learners $\mathcal{H}$ by risk map (4) and randomly initialise the weights $w_{ki}$
2. Repeat for $t = 1, ..., T$:
3.   Repeat for $k = 1, ..., K$:
4.     Find weak-learners $h_{kt}$ that maximise $\sum_i w_{ki} \cdot \mathrm{h}_{kt}(\mathbf{x}_i), \mathrm{h}_{kt} \in \mathcal{H}$.
5.     Find the weak-learner weights $\alpha_{kt}$ that maximise $\mathrm{J}(\mathrm{H} + \alpha_{kt}\mathrm{h}_{kt})$.
6.     Update the weights by $w_{ki} = \frac{y_i - P(\mathbf{x}_i)}{P(\mathbf{x}_i)} \cdot P_k(\mathbf{x}_i)$.
7.   End
8. End
---

Figure 4: **Pseudocode of MCBoost algorithm**

at $t$-th round of boosting, to maximise

$$\sum_i w_{ki} \cdot \mathrm{h}_{kt}(\mathbf{x}_i), \qquad \mathrm{h}_{kt} \in \mathcal{H}, \tag{3}$$

where $\mathrm{h}_{kt} \in \{-1, +1\}$ and $\mathcal{H}$ is a reduced set of weak-learners for speeding up the proposed multiple classifier boosting. The reduced set is obtained by restricting the location of weak-learners around the expected decision boundary. Each weak-learner, $h(\mathbf{x}) = \mathrm{sign}(\mathbf{a}^T\mathbf{x} + b)$, where $\mathbf{a}$ and $b$ represent a simple feature and its threshold respectively, can be represented by $\mathbf{a}^T(\mathbf{x} - \mathbf{x}_o)$, where $\mathbf{x}_o$ is interpreted as the location of the weak-learner. By limiting $\mathbf{x}_o$ to the data points that have high risk to be misclassified, the complexity of searching weak-learners at each round of boosting is greatly reduced. The risk is defined as

$$R(\mathbf{x}_i) = \exp\{-\frac{\sum_{j \in \mathcal{N}_i^B} \|\mathbf{x}_i - \mathbf{x}_j\|^2}{1 + \sum_{j \in \mathcal{N}_i^W} \|\mathbf{x}_i - \mathbf{x}_j\|^2}\} \tag{4}$$

where $\mathcal{N}_i^B$ and $\mathcal{N}_i^W$ are the set of predefined number of nearest neighbors of $\mathbf{x}_i$ in the opposite class and the same class of $\mathbf{x}_i$ (See Figure 3). The weak-learner weights $\alpha_{kt}, k = 1, ..., K$ are then found to maximise $\mathrm{J}(\mathrm{H} + \alpha_{kt}\mathrm{h}_{kt})$ by a line search. Following the AnyBoost method [8], we set the sample weights as the derivative of the cost function with respect to the classifier score. For the cost function $\mathrm{J} = \log \prod_i P(\mathbf{x}_i)^{y_i}(1 - P(\mathbf{x}_i))^{(1-y_i)}$, where $y_i \in \{0, 1\}$ is the label of $i$-th sample, the weight of $k$-th classifier over $i$-th sample is updated by

$$w_{ki} = \frac{\partial \mathrm{J}}{\partial \mathrm{H}_k(\mathbf{x}_i)} = \frac{y_i - P(\mathbf{x}_i)}{P(\mathbf{x}_i)} \cdot P_k(\mathbf{x}_i). \tag{5}$$

See Figure 4 for the pseudocode of the proposed method.

## 3.1 Data clustering

We propose a new data clustering method which assigns a positive sample $\mathbf{x}_i$ to a classifier (or cluster) that has the highest $P_k(\mathbf{x}_i)$.

The sample weight of $k$-th classifier in (5) is determined by the joint probability $P(\mathbf{x})$ and the probability of $k$-th classifier $P_k(\mathbf{x})$. For a negative class ($y_i = 0$), the weights only depend on the probability of $k$-th classifier. The classifier gives high weights to the negative samples that are mis-classified by itself, independently of other classifiers. For a positive class, high weights are assigned to the samples that are misclassified jointly (i.e. the left term in (5)) but may be correctly classified by the $k$-th classifier at next rounds (i.e. high $P_k(\mathbf{x})$). That is, classifiers concentrate on samples in their expertise through the rounds of boosting. This can be interpreted as data partitioning.

## 3.2 Examples

Figure 3 (right) illustrates the concept of the MCBoost algorithm. The method iterates two main steps: learning weak-learners and updating sample weights. States in the figure represent the sam-

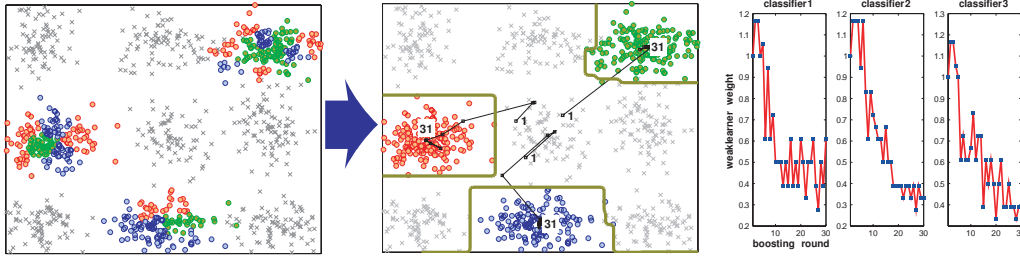

Figure 5: **Example of learning on XOR classification problem.** *For a given random initialisation (three different color blobs in the left), the method learns three classifiers that nicely settle into desired clusters and decision boundaries (middle). The weak-learner weights (right) show the convergence.*

ples that are correctly classified by weak-learners at each step. The sample weighting (5) is represented by data re-allocation. Assume that a positive class has samples of three target clusters denoted by $A, B$ and $C$. Samples of more than two target clusters are initially assigned to every classifier. Weak-learners are found to classify dominant samples (bold letter) in each classifier (step 1). Classifiers then re-assign samples according to their expertise (step 2): Samples $C$ that are misclassified by all are given more importance (bold letter). Samples $B$ are moved to the third classifier as the expert on $B$. The first classifier learns next weak-learners for classifying sample $C$ while the second and third classifiers focus on samples $A$ and $B$ respectively (step 3). Similarly, samples $A, C$ are moved into the respective most experts (step 4) and all re-allocated samples are correctly classified by weak-learners (step 5).

We present an example of XOR classification problems (See Figure 5). The positive class (circle) comprising the three sub-clusters and the negative class (cross) in background make the XOR configuration. Any single or double boosting classifiers, therefore, cannot successfully dichotomise the classes. We exploit vertical or horizontal lines as weak-learners and set the number of classifiers $K$ to be three. We performed random partitioning of positive samples (shown in the left by three different color blobs) for initialising the sample weights. The final decision boundaries and the tracks of data cluster centres of the three boosting classifiers are shown in the middle. Despite the mixed-up initialisation, the method learns the three classifiers that nicely settle into the target clusters after a bit of jittering in the first few rounds. The weak-learner weights (in the right) show the convergence of the three classifiers. Note that the method does not exploit any distance information between input data points, by which conventional clustering methods can apparently yield the same data clusters in this example. As exemplified in Figure 2, obtaining desired data clusters by conventional ways are, however, difficult in practice. The proposed method works well with random initialisations and desirably exhibits quicker convergence when a better initialisation is given.

### 3.3   Discussion on mixture of experts and future work

The existing local optimisation method, MoE, suffers from the absence of a good initialisation solution, but has nice properties once a good initialisation exists. We have implemented MoE in the Anyboost framework. The sample probability in MoE is

$$P(\mathbf{x}_i) = 1/(1 + \exp(-\sum_k Q_k(\mathbf{x}_i) \cdot \mathrm{H}_k(\mathbf{x}_i)))$$

where $Q_k(\mathbf{x}_i)$ is the responsibility of $k$-th classifier over $\mathbf{x}_i$. Various clustering methods can define the function $Q_k(\mathbf{x}_i)$. By taking the derivative of the cost function, the sample weight of $k$-th classifier is given as $w_{ki} = (y_i - P(\mathbf{x}_i)) \cdot Q_k(\mathbf{x}_i)$. An EM-like algorithm iterates each round of boosting and the update of $Q_k(\mathbf{x}_i)$. Dynamic selection of local experts helps time-efficient classification as it does not use all experts.

Useful future studies on the MCBoost method include development of a method to automatically determine $K$, the number of classifiers. At the moment, we first try a large $K$ and decide the right number as the number of visually heterogeneous clusters obtained (See Section 4). A post-corrective step of initial weak-learners would be useful for more efficient classification. When the classifiers start from wrong initial clusters and oscillate between clusters until settling down, some initial weak-

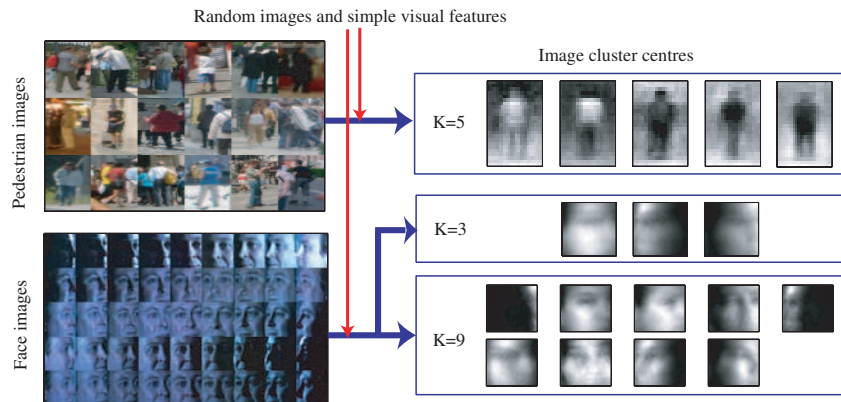

Figure 6: **Perceptual clusters of pedestrian and face images.** *Clusters are found to maximise discrimination power of pedestrian and face images from random images by simple visual features.*

learners are wrong and others may be wasted to make up for the wrong ones. Once the classifiers find right clusters, they exhibit convergence by decreasing the weak-learner weights.

## 4 Experiments

We performed experiments using a set of INRIA pedestrian data [10] and PIE face data [9]. The INRIA set contains 618 pedestrian images as a positive class and 2436 random images as a negative class in training and 589 pedestrian and 9030 random images in testing. The pedestrian images show wide-variations in background, human pose and shapes, clothes and illuminations (Figure 6). The PIE data set involves 900 face images as a positive class (20 persons, 9 poses and 5 lighting conditions) and 2436 random images as a negative class in training and 900 face and 12180 random images in testing. The 9 poses are distributed form left profile to right profile of face, and the 5 lighting conditions make sharp changes on face appearance as shown in Figure 6. Some facial parts are not visible depending on both pose and illumination. All images are cropped and resized into $24 \times 24$ pixel images. A total number of 21780 simple rectangle features (as shown in Figure 1) were exploited.

MCBoost learning was performed with the initial weights that were obtained by the k-means clustering method. Avoiding the case that any of the k-means clusters is too small (or zero) in size has helped quick convergence in the proposed method. We set the portion of high risk data as 20% of total samples for speeding up. The number of classifiers was set as $K \in \{2, 3, 4, 5\}$ and $K \in \{3, 5, 7, 9\}$ for the INRIA and PIE data set respectively. For all cases, every classifier converged within 50 boosting rounds.

Figure 6 shows the cluster centers obtained by the proposed method. The object images were partitioned into $K$ clusters (or classifiers) by assigning them to the classifier that has the highest $P_k(\mathbf{x})$. For the given pedestrian images, the first three cluster centres look unique and the last two are rather redundant. The three pedestrian clusters obtained are intuitive. They emphasise the direction of intensity changes at contours of the human body as discriminating cues of pedestrian images from random images. It is interesting to see distinction of upper and lower body in the second cluster, which may be due to different clothes. For the PIE data set, the obtained face clusters reflect both pose and illumination changes, which is somewhat different from our initial expectation of getting purely pose-wise clusters as the case in Figure 1. This result is, however, also reasonable when considering the strong illumination conditions that cause shadowing of face parts. For example, frontal faces whose right-half side is not visible by the lighting cannot share any features with those having left-half side not visible. Certain profile faces rather share more facial features (e.g. one eye, eye brow and a half mouth) with the half-shadowed frontal faces, jointly making a cluster. All 9 face clusters seem to capture unique characteristics of the face images.

We have also evaluated the proposed method in terms of classification accuracy. Figure 7 shows false-negative and false-positive curves of MCBoost method and AdaBoost method [7]. We set all

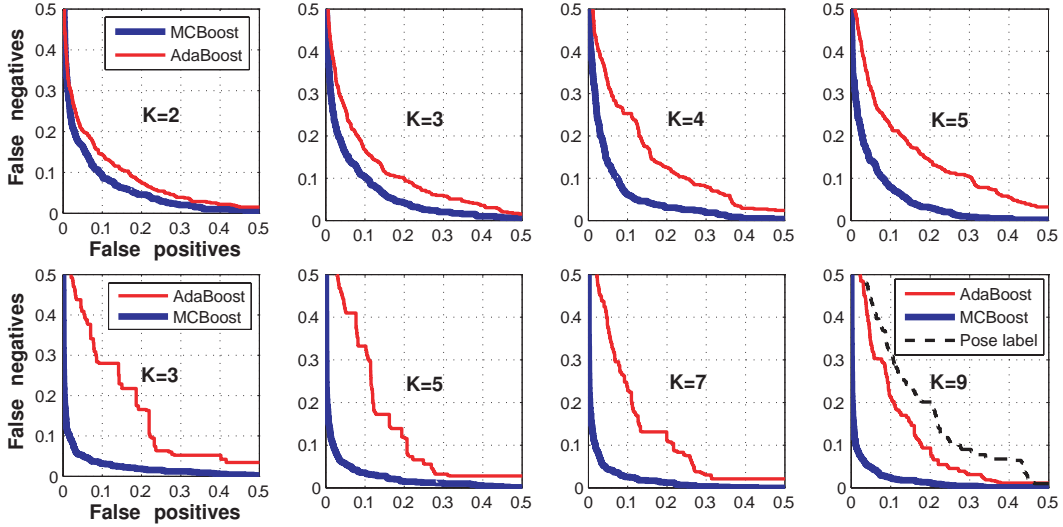

Figure 7: **ROC curves for the pedestrian data (top four) and face data (bottom four).** *MCBoost significantly outperformed AdaBoost method for both data sets and different cluster numbers $K$. MCBoost is also much superior to AdaBoost method learnt with manual pose label (bottom right).*

conditions (e.g. number of weak-learners) equivalent in both methods. The k-means clustering method was applied to positive samples. Boosting classifiers were individually learnt by the positive samples of each cluster and all negative samples in AdaBoost method. The clusters obtained by the k-means method were exploited as the initialisation in MCBoost method. For the PIE data set, we also performed data partitioning by the manual pose label and learnt boosting classifiers separately for each pose in AdaBoost method. For both pedestrian and face experiments and all different number of classifiers $K$, MCBoost significantly outperformed AdaBoost method by finding optimal data clusters and associated feature sets. Our method is also much superior to the Adaboost learnt with manual pose labels (bottom right).

In the AdaBoost method, increasing number of clusters deteriorated the accuracy for the pedestrian data, whereas it increased the performance for the face data. This may be explained by the number of meaningful data clusters. We observed in Figure 6 that there are only three heterogenous pedestrian clusters while there are more than nine face clusters. In general, a smaller number of positive samples in each classifier (i.e. a larger $K$) causes performance degradation, if it is not counteracted

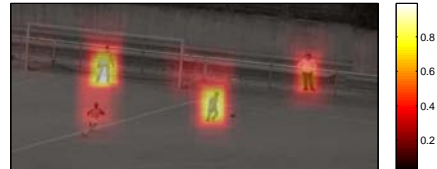

Figure 8: **Example pedestrian detection result.**

by finding meaningful clusters. We deduce, by a similar reason, that the performance of our method was not much boosted when the number of classifiers was increased (although it tended to gradually improve the accuracy for both data sets).

Figure 8 shows an example pedestrian detection result. Scanning the example image yields a total number of 172,277 image patches to classify. Our method ran in 3.6 seconds by non-optimised Matlab codes in a 3GHz CPU PC.

## 5 Conclusions

We have introduced a discriminative co-clustering problem of images and visual features and have proposed a method of multiple classifier boosting called MCBoost. It simultaneously learns image clusters and boosting classifiers, each of which has expertise on an image cluster. The method works well with either random initialisation or initialisation by conventional unsupervised clustering

methods. We have shown in the experiments that the proposed method yields perceptual co-clusters of images and features. In object detection tasks, it significantly outperforms two conventional designs that individually learn multiple boosting classifiers by the clusters obtained by the k-means clustering method and pose-labels.

We will apply MCBoost to various other co-clustering problems in the future. Some useful studies on MCBoost method have also been discussed in Section 3.3. Learning with a more exhaustive training set would improve the performance of the method in object detection tasks.

**Acknowledgements**

The authors are grateful to many people who have helped by proofreading drafts and providing comments and suggestions. They include Z. Ghahramani, B. Stenger, T. Woodley, O. Arandjelovic, F. Viola and J. Kittler. T-K. Kim is financially supported by the research fellowship of the Sidney Sussex College of the University of Cambridge.

# References

[1] I.S. Dhillon, S. Mallela and D.S. Modha, Information-theoretic co-clustering, *Proc. ACM SIGKDD Int'l Conf. on Knowledge discovery and data mining*, pages 89–98, 2003.

[2] M.I. Jordan and R.A. Jacobs, Hierarchical mixture of experts and the EM algorithm, *Neural Computation*, 6(2):181–214, 1994.

[3] P. Viola and M. Jones, Robust real-time object detection, *Int'l J. Computer Vision*, 57(2):137–154, 2002.

[4] A. Torralba, K. P. Murphy and W. T. Freeman, Sharing visual features for multiclass and multiview object detection, *IEEE Trans. on Pattern Analysis and Machine Intelligence*, 29(5):854–869, 2007.

[5] P. Viola, J.C. Platt and C. Zhang, Multiple Instance Boosting for Object Detection, *Proc. Advances in Neural Information Processing Systems*, pages 1417–1426, 2006.

[6] S.Z. Li and Z. Zhang, Floatboost learning and statistical face detection, *IEEE Trans. on Pattern Analysis and Machine Intelligence*, 26(9):1112–1123, 2004.

[7] R. Schapire, The strength of weak learnability, *Machine Learning*, 5(2):197–227, 1990.

[8] L. Mason, J. Baxter, P. Bartlett and M. Frean, Boosting algorithms as gradient descent, *Proc. Advances in Neural Information Processing Systems*, pages 512–518, 2000.

[9] T. Sim, S. Baker, and M. Bsat, The CMU Pose, Illumination, and Expression Database, *IEEE Trans. on Pattern Analysis and Machine Intelligence*, 25(12):1615–1618, 2003.

[10] N. Dalal and B. Triggs, Histograms of Oriented Gradients for Human Detection, *Proc. IEEE Conf. Computer Vision and Pattern Recognition*, pages 886–893, 2005.

